# Constrained Optimization Applied to the Parameter Setting Problem for Analog Circuits

David Kirk, Kurt Fleischer, Lloyd Watts,* Alan Barr
Computer Graphics 350-74
California Institute of Technology
Pasadena, CA 91125

## Abstract

We use constrained optimization to select operating parameters for two circuits: a simple 3-transistor square root circuit, and an analog VLSI artificial cochlea. This automated method uses computer controlled measurement and test equipment to choose chip parameters which minimize the difference between the actual circuit's behavior and a specified goal behavior. Choosing the proper circuit parameters is important to compensate for manufacturing deviations or adjust circuit performance within a certain range. As biologically-motivated analog VLSI circuits become increasingly complex, implying more parameters, setting these parameters by hand will become more cumbersome. Thus an automated parameter setting method can be of great value [Fleischer 90]. Automated parameter setting is an integral part of a *goal-based engineering design methodology* in which circuits are constructed with parameters enabling a wide range of behaviors, and are then "tuned" to the desired behaviors automatically.

## 1 Introduction

Constrained optimization methods are useful for setting the parameters of analog circuits. We present two experiments in which an automated method successfully finds parameter settings which cause our circuit's behavior to closely approximate the desired behavior. These parameter-setting experiments are described in Section 3. The difficult subproblems encountered were (1) building the electronic setup

to acquire the data and control the circuit, and (2) specifying the computation of deviation from desired behavior in a mathematical form suitable for the optimization tools. We describe the necessary components of the electronic setup in Section 2, and we discuss the selection of optimization technique toward the end of Section 3.

Automated parameter setting can be an important component of a system to build accurate analog circuits. The power of this method is enhanced by including appropriate parameters in the initial design of a circuit: we can build circuits with a wide range of behaviors and then "tune" them to the desired behavior. In Section 6, we describe a comprehensive design methodology which embodies this strategy.

## 2    Implementation

We have assembled a system which allows us to test these ideas. The system can be conceptually decomposed into four distinct parts:

**circuit:** an analog VLSI chip intended to compute a particular function.

**target function:** a computational model quantitatively describing the desired behavior of the circuit. This model may have the same parameters as the circuit, or may be expressed in terms of biological data that the circuit is to mimic.

**error metric:** compares the target to the actual circuit function, and computes a difference measure.

**constrained optimization tool:** a numerical analysis tool, chosen based on the characteristics of the particular problem posed by this circuit.

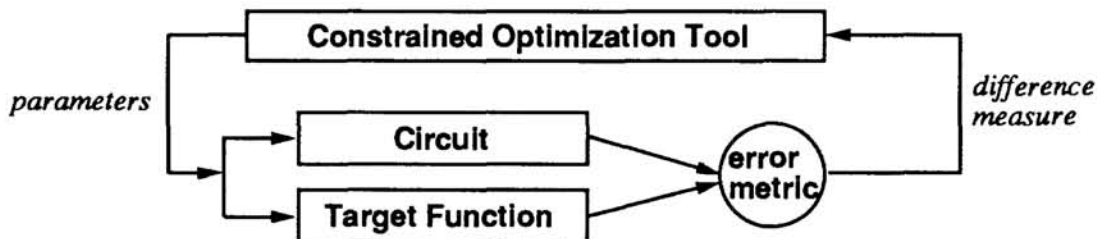

The constrained optimization tool uses the error metric to compute the difference between the performance of the circuit and the target function. It then adjusts the parameters to minimize the error metric, causing the actual circuit behavior to approach the target function as closely as possible.

### 2.1    A Generic Physical Setup for Optimization

A typical physical setup for choosing chip parameters under computer control has the following elements: an analog VLSI circuit, a digital computer to control the optimization process, computer programmable voltage/current sources to drive the chip, and computer programmable measurement devices, such as electrometers and oscilloscopes, to measure the chip's response.

The combination of all of these elements provides a self-contained environment for testing chips. The setting of parameters can then be performed at whatever level

of automation is desirable. In this way, *all* inputs to the chip and all measurements of the outputs can be controlled by the computer.

## 3    The Experiments

We perform two experiments to set parameters of analog VLSI circuits using constrained optimization. The first experiment is a simple one-parameter system, a 3-transistor "square root" circuit. The second experiment uses a more complex time-varying multi-parameter system, an analog VLSI electronic cochlea. The artificial cochlea is composed of cascaded 2nd order section filters.

### 3.1    Square Root Experiment

In the first experiment we examine a "square-root" circuit [Mead 89], which actually computes $ax^\alpha + b$, where $\alpha$ is typically near 0.4. We introduce a parameter (V) into this circuit which varies $\alpha$ indirectly. By adjusting the voltage V in the square root circuit, as shown in Figure 1(a), we can alter the shape of the response curve.

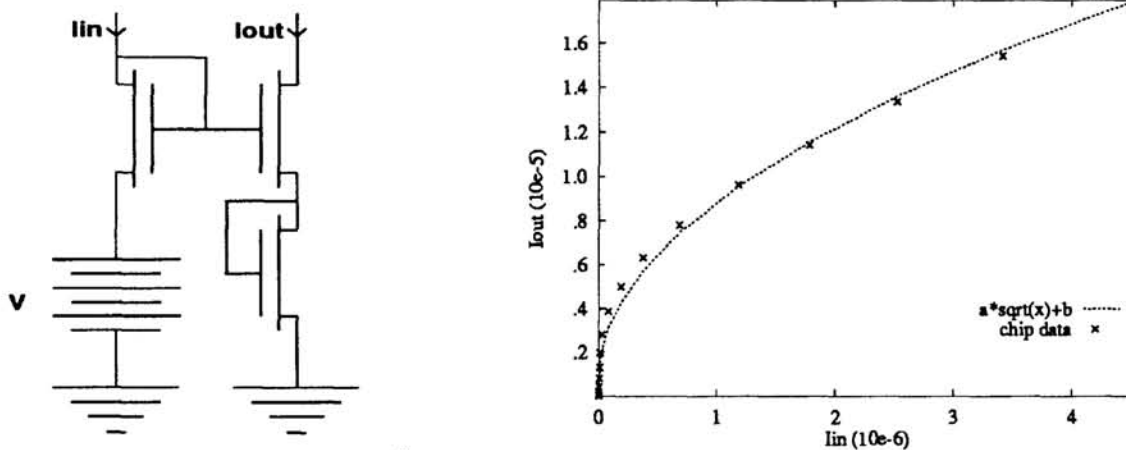

Figure 1: (a) Square root circuit. (b) Resulting fit.

We have little control over the values of $a$ and $b$ in this circuit, so we choose an error metric which optimizes $\alpha$, targeting a curve which has a slope of 0.5 in log-log $I_{in}$ vs. $I_{out}$ space. Since $b << a\sqrt{x}$, we can safely ignore $b$ for the purposes of this parameter-setting experiment. The entire optimization process takes only a few minutes for this simple one-parameter system. Figure 1(b) shows the final results of the square root computation, with the circuit output normalized by $a$ and $b$.

### 3.2    Analog VLSI Cochlea

As an example of a more complex system on which to test the constrained optimization technique, we chose a silicon cochlea, as described by [Lyon 88]. The silicon cochlea is a cascade of lowpass second-order filter sections arranged such that the natural frequency $\tau$ of the stages decreases exponentially with distance into the

cascade, while the quality factor Q of the filters is the same for each section (tap). The value of Q determines the peak gain at each tap.

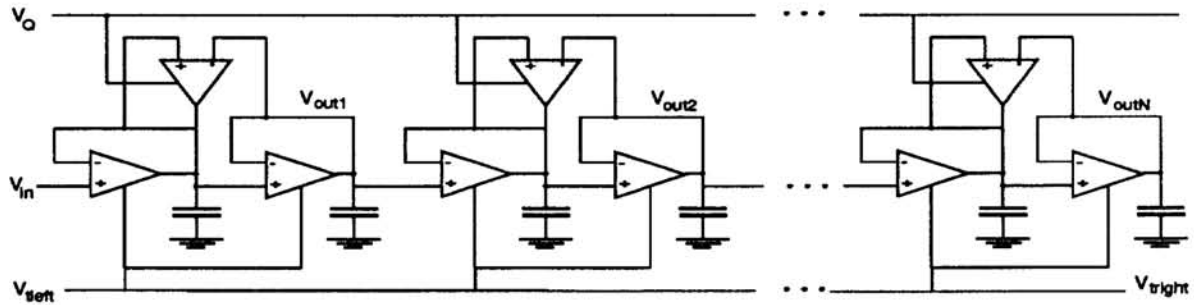

Figure 2: Cochlea circuit

To specify the performance of such a cochlea, we need to specify the natural frequencies of the first and last taps, and the peak gain at each tap. These performance parameters are controlled by bias voltages $V_{T_L}$, $V_{T_R}$, and $V_Q$, respectively. The parameter-setting problem for this circuit is to find the bias voltages that give the desired performance. This optimization task is more lengthy than the square root optimization. Each measurement of the frequency response takes a few minutes, since it is composed of many individual instrument readings.

### 3.2.1   Cochlea Results

The results of our attempts to set parameters for the analog VLSI cochlea are quite encouraging.

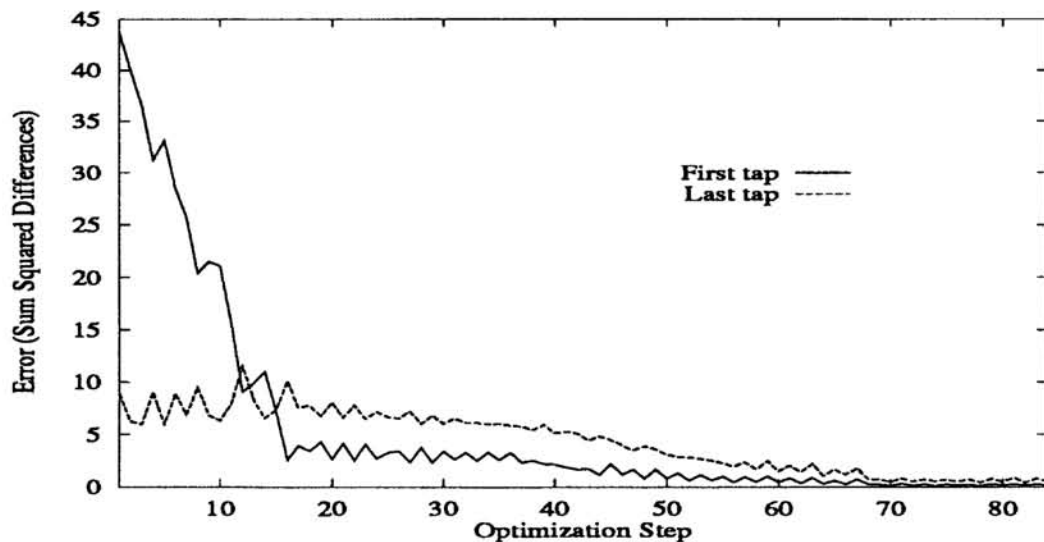

Figure 3: Error metric trajectories for gradient descent on cochlea

Figure 3 shows the trajectories of the error metrics for the first and last tap of the cochlea. Most of the progress is made in the early steps, after which the optimization

is proceeding along the valley of the error surface, shown in Figure 5.

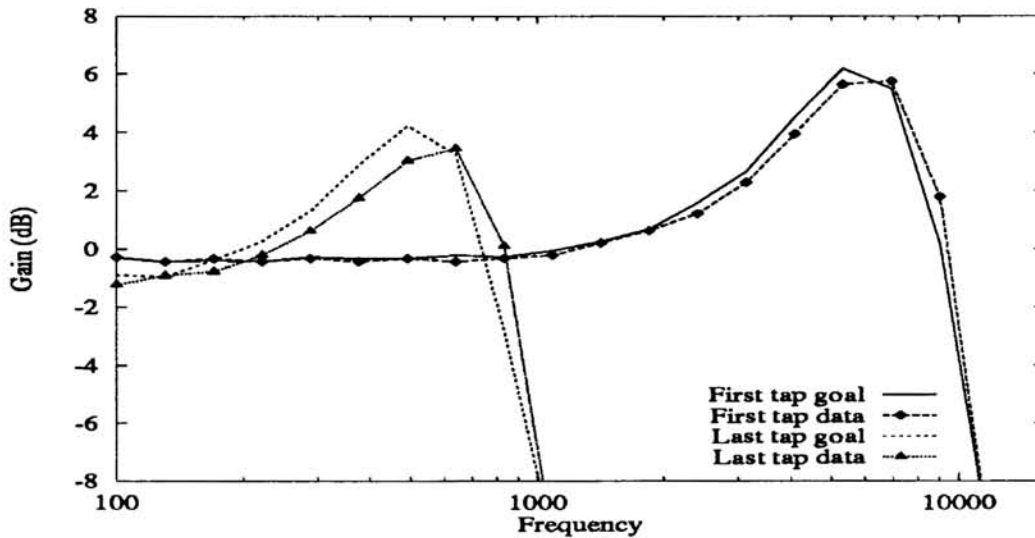

Figure 4: Target frequency response and gradient descent optimized data for cochlea

Figure 4 shows both the target frequency response data and the frequency responses which result from our chosen parameter settings. The curves are quite similar, and the differences are at the scale of measurement noise and instrument resolution in our system.

### 3.2.2  Cochlea Optimization Strategies

We explored several optimization strategies for finding the best parameters for the electronic cochlea. Of these, two are of particular interest:

**special knowledge:** use a priori knowledge of the effect of each knob to guide the optimization

**gradient descent:** assume that we know nothing except the input/output relation of the chip. Then we can estimate the gradient for gradient descent by varying the inputs. Robust numerical techniques such as conjugate gradient can also be helpful when the energy landscape is steep.

We found the gradient descent technique to be reliable, although it did not converge nearly as quickly as the "special knowledge" optimization. This corresponds with our intuition that any special knowledge we have about the circuit's operation will aid us in setting the parameters.

## 4   Choosing An Appropriate Optimization Method

One element of our system which has worked without much difficulty is the optimization. However, more complex circuits may require more sophisticated optimization methods. A wide variety of constrained optimization algorithms exist which are

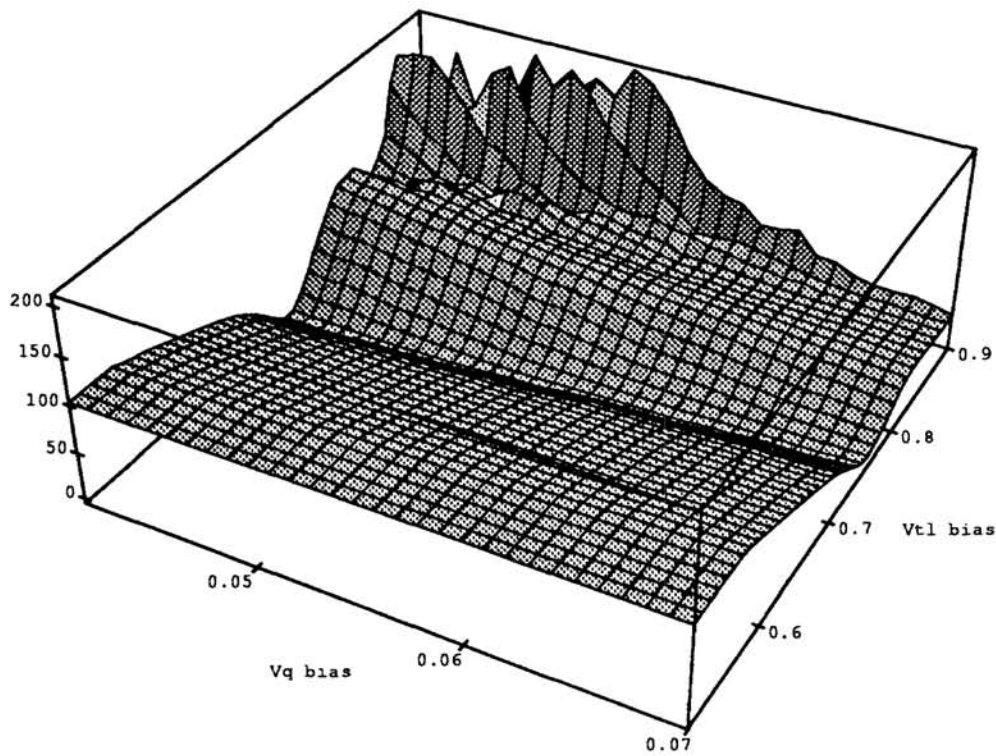

Figure 5: The error surface for the error metric for the frequency response of the first tap of the cochlea. Note the narrow valley in the error surface. Our target (the minimum) lies near the far left, at the deepest part of the valley.

effective on particular classes of problems (gradient descent, quasi-newton, simulated annealing, etc) [Platt 89, Gill 81, Press 86, Fleischer 90], and we can choose a method appropriate to the problem at hand. Techniques such as simulated annealing can find optimal parameter combinations for multi-parameter systems with complex behavior, which gives us confidence that our methods will work for more complex circuits.

The choice of error metric may also need to be reconsidered for more complex circuits. For systems with time-varying signals, we can use an error metric which captures the time course of the signal. We can deal with hysteresis by beginning at a known state and following the same path for each optimization step. Noisy and non-smooth functions can be improved by averaging data and using robust numeric techniques which are less sensitive to noise.

## 5  Conclusions

The constrained optimization technique works well when a well-defined goal for chip operation can be specified. We can compare automated parameter setting with adjustment by hand: consider that humans often fail in the same situations where optimization fails (eg. multiple local minima). In contrast, for larger dimensional spaces, hand adjustment is very difficult, while an optimization technique may succeed. We expect to integrate the technique into our chip development process, and future developments will move the optimization and learning process gradually into the chip. It is interesting to note that our gradient descent method "learns" the parameters of the chip in a manner similar to backpropagation. Seen from this

perspective, this work is a step on the path toward robust on-chip learning.

In order to use this technique, there are two moderately difficult problems to address. First, one must assemble and interface the equipment to set parameters and record results from the circuit under computer control (eg. voltage and current sources, electrometer, digital oscilloscope, etc). This is a one-time cost since a similar setup can be used for many different circuits. A more difficult issue is how to specify the target function of a circuit, and how to compute the error metric. For example, in the simple square-root circuit, one might be more concerned about behavior in a particular region, or perhaps along the entire range of operation. Care must be taken to ensure that the combination of the target model and the error metric accurately describes the desired behavior of the circuit.

The existence of an automated parameter setting mechanism opens up a new avenue for producing accurate analog circuits. The goal of *accurately* computing a function differs from the approach of providing a cheap (simple) circuit which loosely *approximates* the function [Gilbert 68] [Mead 89]. By providing appropriate parameters in the design of a circuit, we can ensure that the desired function is in the domain of possible circuit behaviors (given expected manufacturing tolerances). Thus we define the domain of the circuit in anticipation of the parameter setting apparatus. The optimization methods will then be able to find the best solution in the domain, which could potentially be accurate to a high degree of precision.

## 6    The Goal-based Engineering Design Technique

The results of our optimization experiments suggest the adoption of a comprehensive *Goal-based Engineering Design Technique* that directly affects how we design and test chips.

Our results change the types of circuits we will try to build. The optimization techniques allow us to aggresively design and build ambitious circuits and more frequently have them work as expected, meeting our design goals. As a corollary, we can confidently attack larger and more interesting problems.

The technique is composed of the following four steps:

1) **goal-setting:** identify the target function, or behavioral goals, of the design
2) **circuit design:** design the circuit with "knobs" (adjustable parameters) in it, attempting to make sure desired (target) circuit behavior is in gamut of the actual circuit, given expected manufacturing variation and device characteristics.
3) **optimization plan:** devise optimization strategy to explore parameter settings. This includes capabilities such as a digital computer to control the optimization, and computer-driven instruments which can apply voltages/currents to the chip and measure voltage/current outputs.
4) **optimization:** use optimization procedure to select parameters to minimize deviation of actual circuit performance from the target function the optimization may make use of *special knowledge* about the circuit, such as "I know that this knob has effect $x$," or interaction, such as "I know that this is a good region, so explore here."

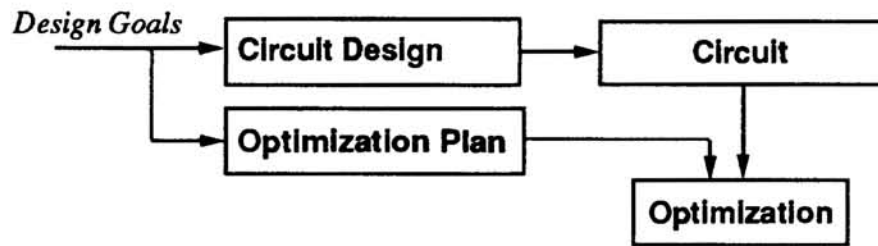

The goal-setting process produces design goals that influence both the circuit design and the form of the optimization plan. It is important to produce a match between the design of the circuit and the plan for optimizing its parameters.

## Acknowledgements

Many thanks to Carver Mead for ideas, encouragement, and support for this project. Thanks also to John Lemoncheck for help getting our physical setup together. Thanks to Hewlett-Packard for equipment donation. This work was supported in part by an AT&T Bell Laboratories Ph.D. Fellowship. Additional support was provided by NSF (ASC-89-20219). All opinions, findings, conclusions, or recommendations expressed in this document are those of the author and do not necessarily reflect the views of the sponsoring agencies.

## Footnotes

*Dept of Electrical Engineering 116-81

## References

[Fleischer 90] Fleischer, K., J. Platt, and A. Barr, "An Approach to Solving the Parameter Setting Problem," IEEE/ACM 23rd Intl Conf on System Sciences, January 1990.

[Gilbert 68] Gilbert, B., "A Precise Four-Quadrant Multiplier with Sub-nanosecond Response," *IEEE Journal of Solid-State Circuits*, SC-3:365, 1968.

[Gill 81] Gill, P. E., W. Murray, and M. H. Wright, "Practical Optimization," Academic Press, 1981.

[Lyon 88] Lyon, R. A., and C. A. Mead, "An Analog Electronic Cochlea," IEEE Trans. Acous. Speech, and Signal Proc., Volume 36, Number 7, July, 1988, pp. 1119-1134.

[Mead 89] Mead, C. A., "Analog VLSI and Neural Systems," Addison-Wesley, 1989.

[Platt 89] Platt, J. C., "Constrained Optimization for Neural Networks and Computer Graphics," Ph.D. Thesis, California Institute of Technology, Caltech-CS-TR-89-07, June, 1989.

[Press 86] Press, W., Flannery, B., Teukolsky, S., Vetterling, W., "Numerical Recipes: the Art of Scientific Computing," Cambridge University Press, Cambridge, 1986.